# Neural Embeddings Rank: Aligning 3D latent dynamics with movements

**Chenggang Chen, Zhiyu Yang, Xiaoqin Wang**
Department of Biomedical Engineering, Johns Hopkins University
`cheng-gang.chen@jhu.edu`

## Abstract

Aligning neural dynamics with movements is a fundamental goal in neuroscience and brain-machine interfaces. However, there is still a lack of dimensionality reduction methods that can effectively align low-dimensional latent dynamics with movements. To address this gap, we propose Neural Embeddings Rank (NER), a technique that embeds neural dynamics into a 3D latent space and contrasts the embeddings based on movement ranks. NER learns to regress continuous representations of neural dynamics (i.e., embeddings) on continuous movements. We apply NER and six other dimensionality reduction techniques to neurons in the primary motor cortex (M1), dorsal premotor cortex (PMd), and primary somatosensory cortex (S1) as monkeys perform reaching tasks. Only NER aligns latent dynamics with both hand position and direction, visualizable in 3D. NER reveals consistent latent dynamics in M1 and PMd across sixteen sessions over a year. Using a linear regression decoder, NER explains 86% and 97% of the variance in velocity and position, respectively. Linear models trained on data from one session successfully decode velocity, position, and direction in held-out test data from different dates and cortical areas (64%, 88%, and 90%). NER also reveals distinct latent dynamics in S1 during consistent movements and in M1 during curved reaching tasks. The code is available at https://github.com/NeuroscienceAI/NER.

## 1 Introduction

It has long been thought that individual neurons in the motor and premotor cortex, similar to those in the sensory cortex, are tuned to specific movement parameters such as direction. However, this static and receptive field-based neural representation fails to explain movement trajectories during simple tasks like reaching. Recent studies have found that the activities of multiple simultaneously recorded neurons, which fire spikes in a time-dependent manner, encode reaching movements [10]. Unlike the one-dimensional dynamics from a single neuron, understanding how movements are represented by these high-dimensional neural dynamics is challenging. In systems neuroscience and brain-machine interfaces, there is significant motivation to reduce these high-dimensional neural dynamics to low-dimensional latent dynamics for at least three reasons:

*First, visualizing neural dynamics*. This involves a trade-off between dimensionality and explained variance. To explain a complex reaching task, at least six dimensions are typically required. For example, in an eight-direction center-out reaching task, [11][9][12] selects fifteen dimensions for the PMd, ten for the M1, and eight for the S1. Therefore, we need to further reduce the dimensionality of these "low-dimensional" latent dynamics. Currently, we still lack a method that can directly explain enough variance within three dimensions. *Second, comparing movements-related latent dynamics*. After dimensionality reduction, we can visualize the trajectories of latent dynamics over time. For instance, [5] reveals rotational latent dynamics during reaching tasks using principal component analysis (PCA). [25] found that animals performing similar tasks exhibit similar latent dynamics.

However, these trajectories do not align precisely with the reaching movements: when the hand reaches in eight directions, the trajectories of latent dynamics are neither in eight distinct directions nor well-separated, often appearing entangled. *Third, decoding movements using latent dynamics.* Decoders trained on individual neural activities are commonly used to predict movements [13]. However, a drawback of using individual neural activities is that when the identities of neurons change during long-term recordings, the decoding performance deteriorates [9]. Additionally, it is infeasible to decode movements from different brain areas or animals. Decoders trained on latent dynamics facilitate long-term and cross-animal decoding [9, 25]. Since latent dynamics do not fully capture neural dynamics, decoding performance is often suboptimal with linear decoders, necessitating the use of nonlinear decoders or deep neural networks [25]. Thus, decoding movements using a linear decoder without hyperparameters remains a significant challenge.

As our goal is to extract latent dynamics that are most informative about movements, we chose to train the latent dynamics using movements as the target. Several recent studies have trained latent dynamics to **classify** different movement directions or positions using variational autoencoders (VAE) [32][14][18] or contrastive learning [27][1]. In this paper, *we are inspired by the fact that many features, including movements, are continuous, and that the function of many biological neurons is not classification but regression.* For example, neurons exhibit monotonic tuning to light intensity and sound levels [2][24]. Even for discrete features like faces, face cells exhibit ramp-shaped tuning to different features[8]. Additionally, 75% of faces could be correctly decoded using a linear regression decoder[4]. Thus, we trained latent dynamics to regress movement by minimizing the ranking loss.

We are motivated by the fact that **CEBRA treats continuous labels as many discrete classes, which cannot be well separated in low-dimensional space. These classes are also highly imbalanced, with many more near-zero classes** (Fig. 1a).

Our two main contributions are as follows:

We introduce Neural Embeddings Rank (NER), a dimensionality reduction method that contrasts paired samples in the embedding space and ranks neural embeddings to align them with continuous movement labels (Fig. 1b). *Without introducing additional hyperparameters and using the same inputs, NER addresses the high dimensionality and class imbalance issues present in CEBRA.*

We demonstrate that NER reveals behavior-aligned latent dynamics in all twenty sessions from three different brain areas in three monkeys. The trajectories of latent dynamics vary across different behaviors. We show that all movement parameters (directions, velocities, and positions) can be decoded using linear decoders trained on data from one session, tested on data from different dates over one year, and across different brain areas and hemispheres.

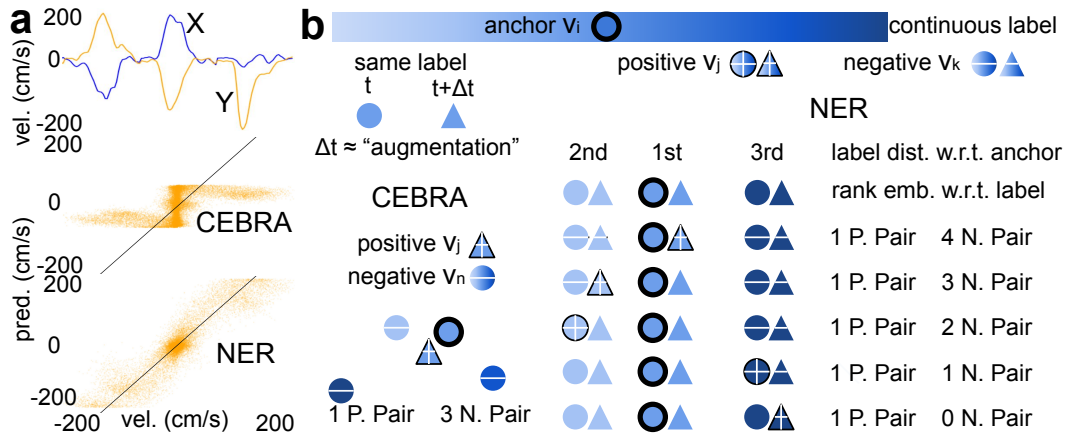

Figure 1: **a** Top: Hand velocities of X-Y coordinates across three trials. Note that the distribution of velocities is highly imbalanced, with many near-zero values. Bottom: Real (X-axis) versus predicted (Y-axis) Y-coordinate hand velocities using a linear regressor. The predicted velocities in CEBRA are much smaller than those in NER, as CEBRA mispredicts infrequent large velocities as frequent small velocities. **b** Batch size is three (in real experiments, it is 512), with two batches (one for $v_i$ and one for $v_j$ and $v_k$) combined in NER. Both models use embeddings within a specific time offset (e.g., 10 ms) of the anchor as the augmented embedding, which shares the same label/color as the anchor.

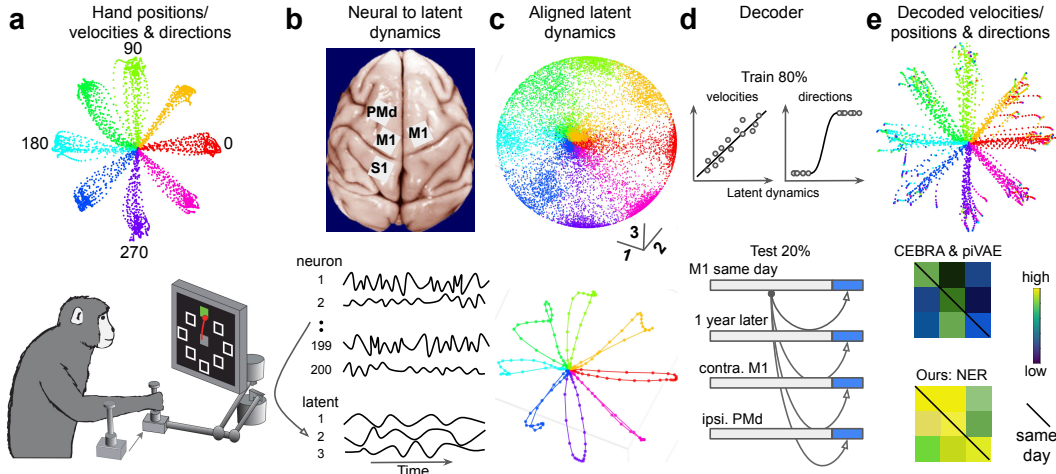

Figure 2: NER aligns 3D latent dynamics with movements and enabling cross sessions movement decoding. **a** Top: The monkey performs a center-out reaching task in eight directions using a planar manipulandum. Hand velocity is computed from hand position. Bottom: The monkey moves the cursor to outer targets to receive rewards (image from [23]). **b** Top: Neural dynamics are recorded using a 96-channel Utah array in two monkeys. Monkey C has implants in the M1 of the right hemisphere first and in M1 and PMd of the left hemisphere second. Monkey L has an implant in area 2 of the S1 of the left hemisphere (image from brainmuseum.org). During the task, all monkeys use the hand contralateral to the implanted hemisphere. Bottom: Spiking activities from multiple neurons (44 to 211) are recorded during multiple trials (168 to 1038) of the behavioral task. Dimensionality reduction reduces 200-dimensional neural dynamics to 3D latent dynamics. **c** Top: Neural dynamics from 190 dimensions (neurons) in the PMd are reduced to 3D latent dynamics. Bottom: Trial-averaged latent dynamics. Data is from Monkey C (date: 2016-10-14). **d** Top: Linear and logistic regression decoders are trained on the same independent variables (latent dynamics) but different dependent variables (hand velocities and directions) using 80% of train data. Bottom: The trained decoder from 80% of train data is used to predict movements on the 20% held-out test data. The model trained on one day predicts movements across one year, in the contralateral M1, and ipsilateral PMd. **e** Top: Two linear decoders trained from **c** decode hand velocities (positions) and directions with R-squared accuracy of 86% (96%) and peak accuracy of 97%, respectively, on the 20% held-out test data. Bottom: Decoders trained on the same day (diagonal) have much better decoding performance than models trained on different conditions (off-diagonal) using previous dimensionality reduction methods, whereas our method has higher and consistent decoding performance.

## 2  Related work

There are at least five categories of dimensionality reduction methods: **Linear** methods, such as PCA, jPCA [5], demixed PCA (dPCA) [16], and preferential subspace identification (PSID) [26]. PCA captures the majority of variance in the data, jPCA reveals rotational dynamics in monkey reaching, dPCA highlights task-related components, and PSID extracts latent dynamics that predict motion during reach versus return epochs. **Nonlinear** methods like uniform manifold approximation and projection (UMAP) [19] and t-distributed stochastic neighbor embedding (tSNE) [28] are extensively used in biological data, such as for identifying neuron cell types [17]. These methods reveal identities but often collapse temporal dynamics that resemble neural trajectories. UMAP with labels has been applied for dimensionality reduction in recent studies [27, 32]. **Generative** methods using recurrent neural networks or Transformers, such as latent factor analysis via dynamical systems (LFADS) [21], AutoLFADS [15], RADICaL [33], and Neural Data Transformer (NDT) [30]. These methods model single-trial variability in neural spiking activity better than PCA, though they often require explicit assumptions about underlying data statistics. **Label-guided generative** methods using VAEs, including Poisson identifiable VAE (piVAE) [32], SwapVAE [18], and targeted neural dynamical modeling (TNDM) [14]. For example, piVAE uses both discrete and continuous labels to shape embeddings, revealing well-separated but less movement-aligned embeddings. **Contrastive** learning methods, introduced to learn robust, generalizable representations of neural population dynamics,

such as CEBRA [27] and Mine Your Own view (MYOW) [1]. Compared to piVAE, AutoLFADS, and UMAP, CEBRA provides more identifiable latent dynamics corresponding to different hand directions in S1, although these latent dynamics trajectories do not correlate well with movements.

## 3 Model

NER is inspired by previous studies [31] and uses the same data sampling and neural feature encoder as CEBRA [27] to extract neural embeddings. Fig. 1b illustrates the difference between NER and CEBRA. CEBRA treats each embedding in a batch as a discrete class. For an anchor, it contrasts with its augmented embedding as a positive pair and three randomly sampled embeddings as negative pairs. NER, on the other hand, ranks six embeddings according to their continuous labels. It contrasts an anchor with its augmented or first embedding as a positive pair and the remaining four embeddings as negative pairs. NER continues by contrasting the second embedding as a positive pair and the remaining three embeddings as negative pairs. This process repeats until all embeddings have been positively contrasted with the anchor. *NER learns a regression-aware representation that orders all embeddings in a batch.*

Mathematically, we define $x$ as the high-dimensional neural dynamics, $f$ as the feature encoder, and $v = f(x)$ as the low-dimensional neural embeddings. The batch size $N$ is set to 512, and the temperature $\tau$ is fixed at 1. Data augmentation in both CEBRA and NER is achieved by selecting embeddings whose labels fall within a specific offset (e.g., 10 ms) of the anchor's label. Note that we did not fine-tune the temperature and offset, as the only difference between CEBRA and NER is the loss function. The selection of these hyperparameters will have a similar effect on both models.

In each iteration, CEBRA receives one batch of anchors $v_i$, one batch of embeddings $v_j$ that will positively contrast with the anchors, and a third batch of randomly sampled embeddings $v_n$ that will negatively contrast with the anchors. The anchor loss $l$ in CEBRA is:

$$l^{(i)}_{CEBRA} = -log \frac{exp(sim(v_i, v_j)/\tau)}{\sum N_{n=1} exp(sim(v_i, v_n)/\tau)}$$

where $sim(\cdot, \cdot)$ represents the similarity between two neural embeddings (e.g., negative $L2$).

In each iteration, NER receives one batch of anchors $v_i$, one batch of augmented embeddings $v_j, v_k$ that will either positively or negatively contrast with an anchor, and a third batch of labels $y$. The anchor loss $l$ in NER is:

$$l^{(i)}_{NER} = \frac{1}{2N-1} \sum 2N_{j=1, j \neq i} - log \frac{exp(sim(v_i, v_j)/\tau)}{\sum_{v_k \in S_{i,j}} exp(sim(v_i, v_k)/\tau)}$$

There are two key differences from CEBRA. First, batches of $v_i$ and $v_j, v_k$ are merged, and labels $y$ are duplicated, resulting in a batch of $2N$ for both embeddings and labels. This ensures that each anchor and its augmented embedding exist within the same batch. Second, we introduce $S_{i,j} := \{v_k \mid k \neq i, d(y_i, y_k) \geq d(y_i, y_j)\}$ to denote the set of embeddings $v_k$ that are of lower ranks than $v_j$ in terms of label distance relative to $v_i$, where $d(\cdot, \cdot)$ is the distance measure between two labels (e.g., $L1$). Intuitively, for an anchor $v_i$, any other embedding $v_j$ in the batch is positively contrasted with it, enforcing the similarity between $v_i$ and $v_j$ to be greater than that between $v_i$ and any other embedding $v_k$ in the batch if the label distance between $y_i$ and $y_k$ is larger than that of $y_i$ and $y_j$. Minimizing $l^{(i)}_{NER}$ aligns the order of embeddings with their corresponding label orders relative to the anchor.

By ranking all the embeddings within a batch, NER also addresses the class imbalance issue and effectively represents infrequent classes (Fig. 1a). In each batch, a small percentage (e.g., 5%) of embeddings may come from infrequent classes, such as large velocities. In CEBRA, only a single augmented embedding is positively contrasted with its anchor, so only 5% of anchors have access to these infrequent embeddings. In NER, all $2N - 1$ embeddings can contrast with an anchor, allowing 100% of anchors to access the 5% of infrequent embeddings.

## 4 Results

To fairly evaluate our dimensionality reduction method against related approaches, we selected six representative methods from various categories: PCA, dPCA, UMAP (with and without labels),

piVAE, and CEBRA. To avoid bias from a single session in a specific brain area, where piVAE and CEBRA were previously tested, we conducted experiments across M1, PMd, and S1, covering a total of twenty sessions (Table 1). Statistical results for each comparison are provided (Table 2). Figure 2 illustrates the pipeline of this study. We primarily used linear decoders, and additionally included nonlinear k-nearest neighbors (kNN) decoder. To maintain consistency with movement decoding in the original CEBRA paper, we used a 16-dimensional (16D) CEBRA model, which has a stronger representation capacity and demonstrates better performance than the 3D model in kNN decoder.

## 4.1 Movement-aligned latent dynamics were consistent over years in M1

Across all ten sessions in the left and right hemispheres of M1, NER consistently revealed neural embeddings that aligned with movement (Fig.3a, Fig.9a). Notably, during the initial movement stage, the latent dynamics converged on the same starting points, forming a pinwheel structure resembling the ground truth movements. Furthermore, we observed nearly identical neural embeddings in both hemispheres, even when data collection was separated by over a year.

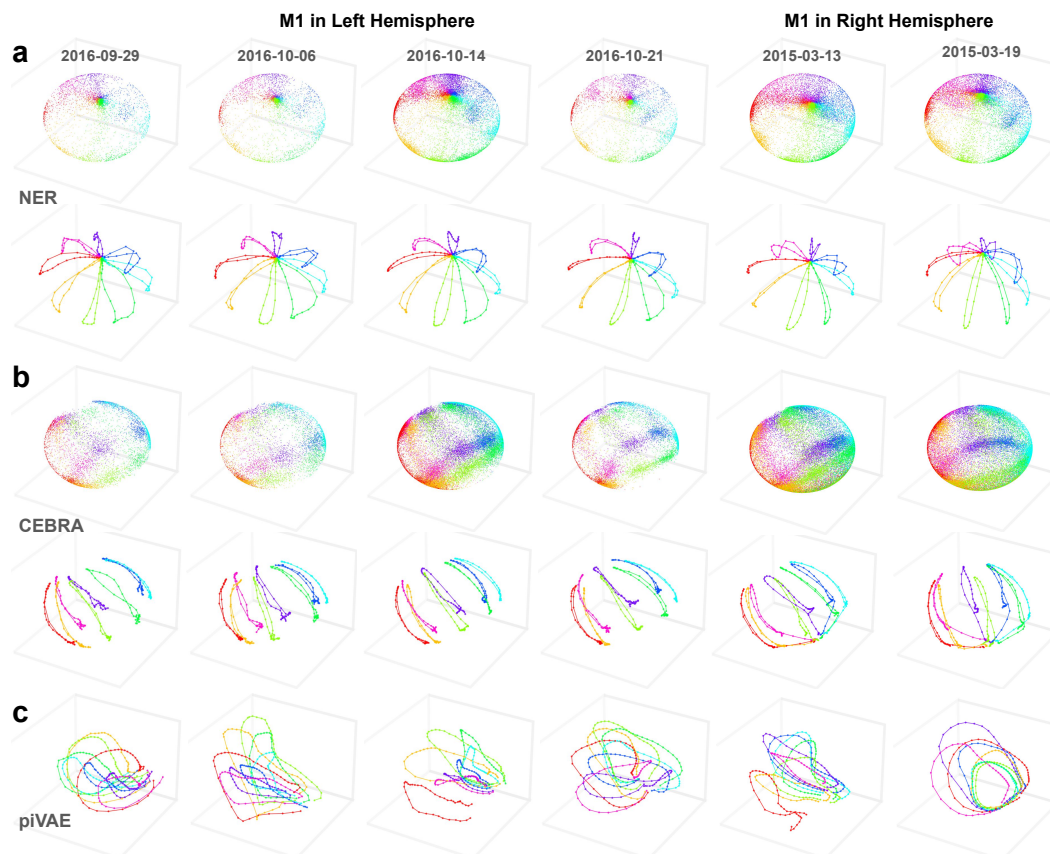

Figure 3: NER reveals consistent, movement-aligned latent dynamics in M1. **a** Single-trial (top) and averaged (bottom) latent dynamics from six sessions across one year in two hemispheres. Latent dynamics are rotated with reference to the 2016-10-14 session, using one of the eight reaching directions. **b-c** Similar to **a** but using CEBRA and piVAE. Fig. 9 shows the latent dynamics from the remaining four sessions for the same monkey. Fig. 10 displays single-trial and/or averaged latent dynamics revealed by five other methods. Fig. 11 shows trial-averaged latent dynamics before rotation. Fig. 12 presents the entangled neural embeddings using PCA and the time-stimulus components revealed by dPCA.

CEBRA was the second-best method, uncovering comparable latent dynamics with both directions and positions roughly aligned with movements for both single and averaged embeddings (Fig.3b, Fig.9b). However, CEBRA had two limitations: the movement starting points were widely separated,

differing from ground truth movements, and its latent dynamics were less consistent across sessions. For example, it only revealed connected latent dynamics at movement starting points in two sessions.

piVAE ranked third, displaying direction-aligned latent dynamics in different directions with relatively separated single-trial neural embeddings (Fig.3c, Fig.9c). However, while correlated with movements, the latent dynamics were not aligned with them and were less consistent across sessions. UMAP with labels showed clearly clustered neural embeddings corresponding to different angles, while UMAP without labels produced extended, less clustered latent dynamics (Fig. 10). Both methods failed to generate aligned and consistent latent dynamics. PCA and dPCA also generated identifiable latent dynamics (Fig.10), with dPCA revealing both time and stimulus components. However, a major limitation of both methods was the mixing of single-trial neural embeddings (Fig.10).

In summary, NER proved to be the best method for revealing movement-aligned latent dynamics. We further evaluated its performance in PMd and S1, leveraging these aligned latent dynamics to decode movements within and across sessions and to explore movement-specific latent dynamics.

## 4.2 Explained variance of movements using linear decoders in M1, PMd, and S1

Five dimensionality reduction methods were used to reveal single-trial latent dynamics that depended on movements. We then used these latent dynamics as independent variables to explain the variance of dependent variables, namely, hand velocities, directions, and positions.

Fig. 4a shows the ground truth and predicted hand movement trajectories using latent dynamics revealed by NER in PMd. A linear regression decoder explained 90% and 98% of the variance in hand velocities and positions, respectively.

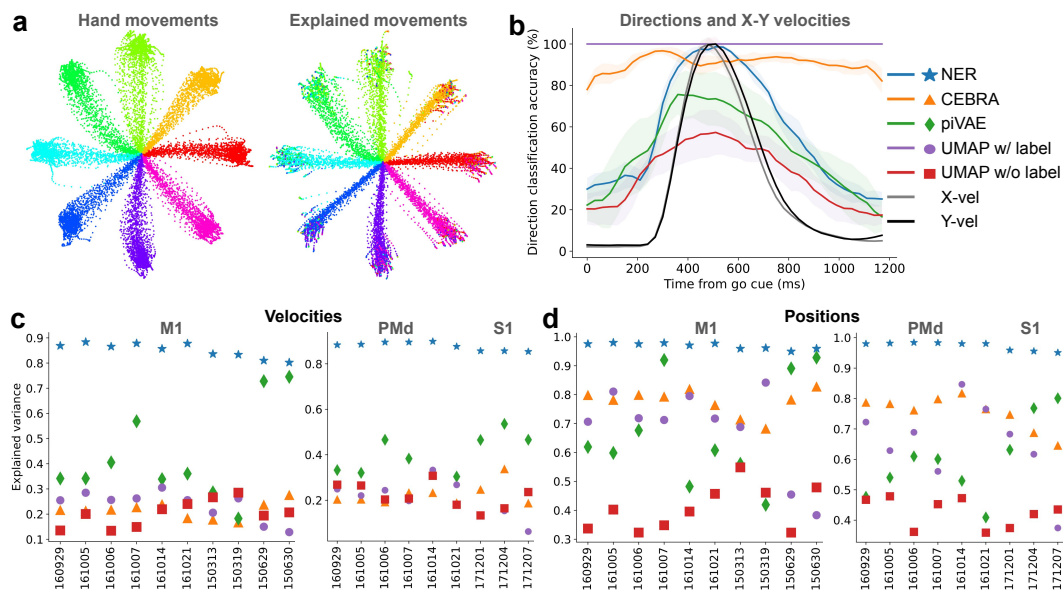

Figure 4: Explained variance of movements in M1, PMd, and S1. **a** Left: Hand movement trajectories. Right: Predicted trajectories by two decoders. Data collected from PMd on 2016-10-14. The explained variance for velocities and positions is 90% and 98%, respectively. **b** Hand direction classification accuracy using a logistic regression decoder. Shaded areas represent the standard deviation across six sessions from M1. **c** Explained variance of hand velocities using a linear regression decoder on latent dynamics revealed by five dimensionality reduction methods (indicated by different colors and shapes). The X-axis shows session dates. Left: Ten sessions from the M1 of Monkey C. Right: Six sessions from PMd of Monkey C and four sessions from S1 of Monkey H. **d** Similar to **c** but for hand positions. Fig. 13 provides the direction tuning curve in PMd, the correlation between tuning curves and velocities, and explained variance for directions.

In both M1 and PMd (Fig.4b, Fig.13a), a logistic regression decoder revealed the tuning of directional accuracy from the start of the go cue to the end of the animal's reach. This tuning curve was highly

correlated with hand velocities in the latent dynamics extracted by NER but not by CEBRA (M1: 0.93 vs. 0.28, PMd: 0.93 vs. 0.13; Fig. 13b).

NER outperformed the four other methods in all sessions for explaining the variance of both hand velocities and positions (Fig. 4c, d). For example, across ten sessions in M1, NER explained 86% of the variance in velocities, while the next best model, piVAE, explained only 35%. Similar findings were observed in PMd (89% vs. 32%) and S1 (86% vs. 47%).

In summary, combined with linear decoders, NER demonstrated the clearest velocity-dependent direction tuning and explained the largest variance in velocities and positions.

### 4.3 Long-term and cross-hemisphere decoding in M1

NER explains the largest variance in both hand velocities and positions in the 80% training data. Next, we tested the trained model on the remaining 20% of test data. In addition to decoding test data from the same session on a single day, we also evaluated its performance on data from different sessions. We first conducted comparisons in M1 (Fig.5), followed by PMd (Fig.6) and S1 (Fig. 7).

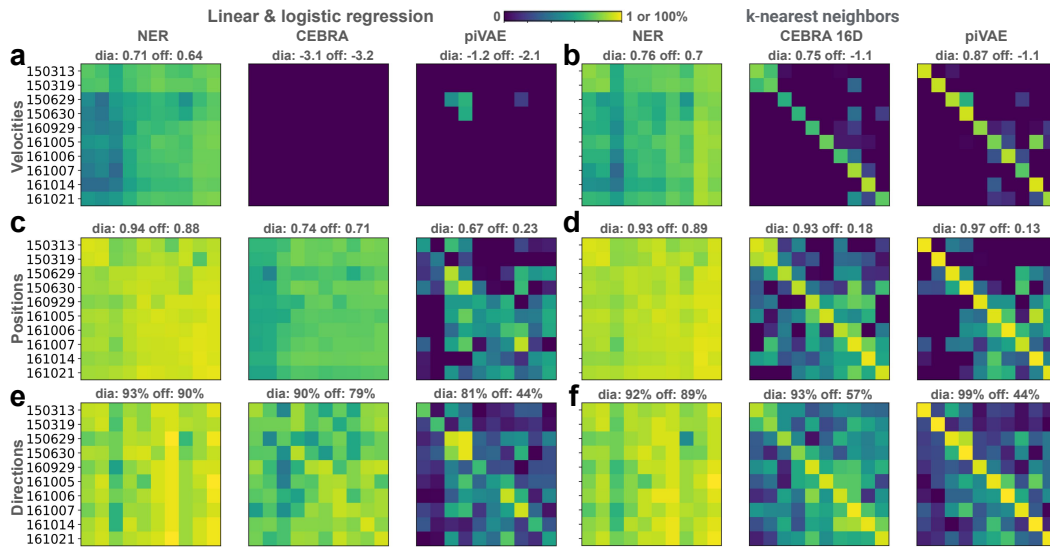

Figure 5: Decoding within and across time and brain hemispheres over years in M1. **a-d** Three methods are applied to neural dynamics from ten sessions. Linear and logistic regression decoders (**a**, **c**, **e**) or a nonlinear k-nearest neighbors (kNN) decoder (**b**, **d**, **f**) are trained on 80% of the data and used to decode velocities, positions, and directions on the remaining 20% within-session (diagonal) and cross-session (off-diagonal). Brighter colors indicate higher decoding performance. **a** Same-session, cross-session, and cross-hemisphere velocity decoding using a linear regression decoder. **b** Similar to **a** but using the nonlinear kNN decoder. Note that CEBRA's latent dynamics are represented in sixteen dimensions instead of three. **c-d** Similar to **a-b** but with hand positions as the decoding variable. **e-f** Similar to **a-b** but with hand directions as the decoding variable.

Fig.5a shows velocity decoding performance using a linear regression decoder across three dimensionality reduction methods. Interestingly, the linear decoder could not decode hand velocities from latent dynamics generated by CEBRA (all variances were negative) and piVAE (only four positive). In contrast, all variances with NER were positive (minimum: 0.24), with performance across different sessions comparable to within-session performance (0.64 vs. 0.71). A kNN decoder (Fig.5b) achieved high performance for CEBRA and piVAE only with within-session latent dynamics.

NER outperformed CEBRA and piVAE by a substantial margin in position decoding across all conditions (Fig.5c). While the kNN decoder did not improve NER's performance relative to the linear decoder, it enhanced within-session performance for CEBRA and piVAE over NER (Fig.5d). However, this came at the cost of cross-session decoding performance (0.89 vs. 0.18 and 0.13). Fig.5e shows direction decoding accuracy, where NER still outperformed CEBRA and piVAE. Similar results were observed using a kNN decoder (Fig.5f).

In summary, compared to CEBRA and piVAE, NER consistently achieved higher performance across all sessions with a linear decoder and outperformed in cross-session decoding with the nonlinear decoder.

## 4.4 Latent dynamics in PMd and decoding between M1 and PMd

Next, we turned our attention to PMd. Surprisingly, NER revealed similar latent dynamics in this higher-order motor area (Fig.6a). Two other dimensionality reduction methods also identified comparable latent dynamics, but these were less consistent and did not align well with movements (Fig.14). Fig. 6b shows within- and cross-session velocity decoding using a linear regression decoder (left) and a kNN decoder (right). Consistent with M1, in PMd, we observed that (1) both CEBRA and piVAE underperformed with linear regression, while NER achieved stable performance across all conditions, regardless of within- or cross-session contexts; (2) NER demonstrated robustness across decoders, whereas the performance of CEBRA and piVAE varied substantially, ranging from very low to occasionally outperforming NER in within-session decoding with the kNN decoder; and (3) even with the kNN decoder, CEBRA and piVAE failed in cross-session decoding, whereas NER maintained similar performance.

We also evaluated position and direction decoding and found that NER continued to outperform the other two methods (Fig.6c, Fig.15). Overall, NER reveals consistent latent dynamics in PMd and can effectively decode movements across PMd and M1.

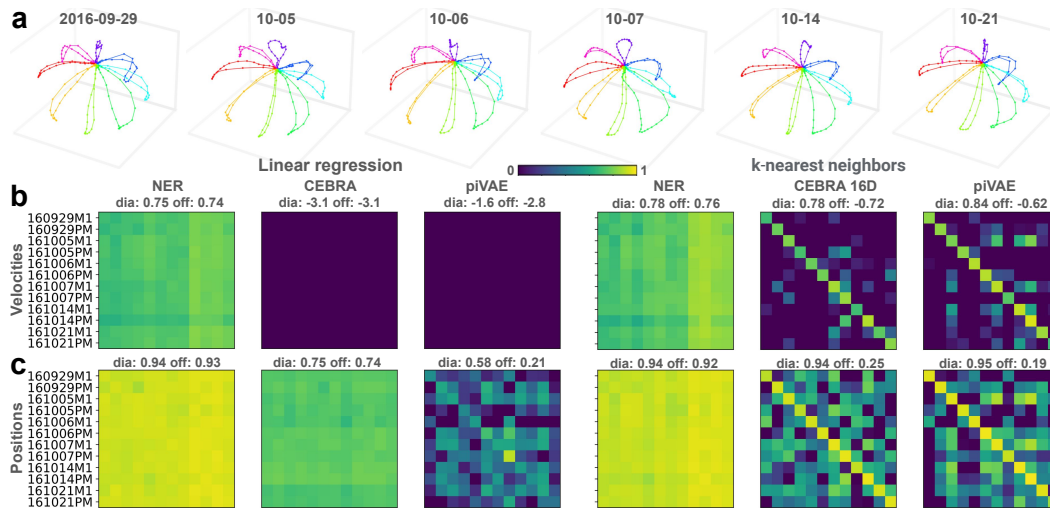

Figure 6: Latent dynamics in PMd and decoding across brain areas. **a** Trial-averaged latent dynamics in PMd revealed by NER, with the rotation reference set to the same session as in Fig. 3 (i.e., 2016-10-14). **b** Previously used latent dynamics from M1 are added. Same-date, cross-date, and cross-brain area velocity decoding using a linear regression decoder (left) and a kNN decoder (right) on latent dynamics revealed by the three methods. **c** Similar to **b**. Fig. 14 shows the single-trial and averaged latent dynamics revealed by CEBRA and piVAE. Fig. 15 shows hand direction decoding performance using a logistic regression decoder.

## 4.5 Same movements but different latent dynamics in S1

Lastly, we examined the latent dynamics and movement decoding in S1. NER revealed consistent latent dynamics in S1 (Fig.7a, b). After rotating the latent dynamics with reference to the target shown in Fig.3, they exhibit a consistent yet distinct shape compared to the latent dynamics observed in M1 and PMd. Velocity decoding using a linear regression decoder was only successful when the latent dynamics were extracted by NER (Fig.7c, d). While all three methods performed adequately for position decoding, NER outperformed both CEBRA and piVAE across all nine conditions (Fig.7e). Although S1 displays different latent dynamics than M1 and PMd, NER remains the most effective method for decoding movement both within and across sessions in S1.

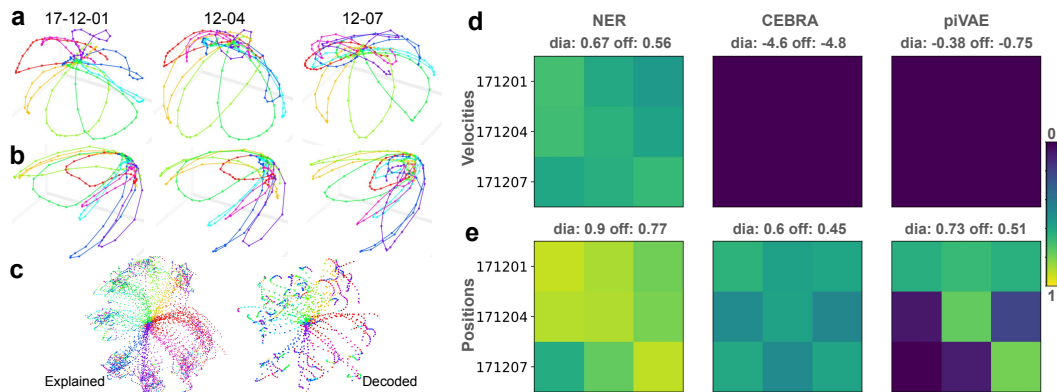

Figure 7: Distinct latent dynamics with the same movement in S1. **a** Trial-averaged latent dynamics revealed by NER, using the same reference target session as in Fig. 3. **b** Latent dynamics with the reference target set to the first session in S1. **c** Left: On 80% held-in trials, the explained variance by linear and logistic regression decoders is 91% and 97%, respectively. Right: On 20% held-out test trials, the trained linear decoder predicts velocities and positions with performances of 71% and 90%, respectively. Data collected on 2017-12-01. **d** Same- and cross-date velocity decoding performance using the linear regression decoder with three dimensionality reduction methods. **e** Similar to **d** but for positions. Fig. 16 shows latent dynamics revealed by CEBRA and piVAE.

## 4.6 Straight and curved hand movements have different latent dynamics in M1

Lastly, we applied both NER and CEBRA in a new experiment where a monkey performed both straight and curved hand movements in different directions while neural recordings were simultaneously collected in M1 (Fig. 8a).

We first examined the latent dynamics when the monkey performed straight hand movements in six different directions (Fig.8b). Surprisingly, both single and averaged latent dynamics aligned well with the movements (Fig.8c) and displayed a shape similar to the previously observed latent dynamics in M1. Next, we selected three hand directions, each involving both straight and curved hand movements (Fig.8e). When we trained both NER and CEBRA on individual directions, only NER revealed clearly separated latent dynamics corresponding to straight and curved movements (Fig.8f). The difference between the two methods became even more pronounced when they were trained on all three directions combined: NER displayed latent dynamics for straight movements that were surrounded by those formed by curved movements. The explained variance achieved by NER was also higher than that of CEBRA across all three angles, especially on the combined angles (Fig. 17c). Finally, we tested a more challenging condition where all six reaching movements were curved (Fig.8g). In the latent space, two curved movements in the same direction produced close but distinct latent dynamics (Fig.8h). While CEBRA performed comparably on single directions, it struggled with combined directions (Fig.17e). NER consistently achieved higher explained variance than CEBRA (Fig.17f).

Overall, NER not only aligns latent dynamics with straight movements but also effectively differentiates curved hand movements from straight ones, demonstrating its robustness across various movement types.

## 5   Discussion

A benchmark comparison of NER and six other dimensionality reduction methods across multiple brain areas and two movement tasks highlights NER's superior performance in uncovering latent dynamics. We believe the primary advantage of our method is its ability to extract nearly identical latent dynamics across brain areas and over extended time periods. This capability opens new avenues for both fundamental neuroscience research and brain-machine interfaces (BMI). Previous studies [9, 25] used PCA to discover preserved latent dynamics across time and in animals performing similar behaviors. In contrast, the latent dynamics revealed by NER are significantly more informative than those uncovered by PCA. We believe NER will aid neuroscientists in probing the stability of latent

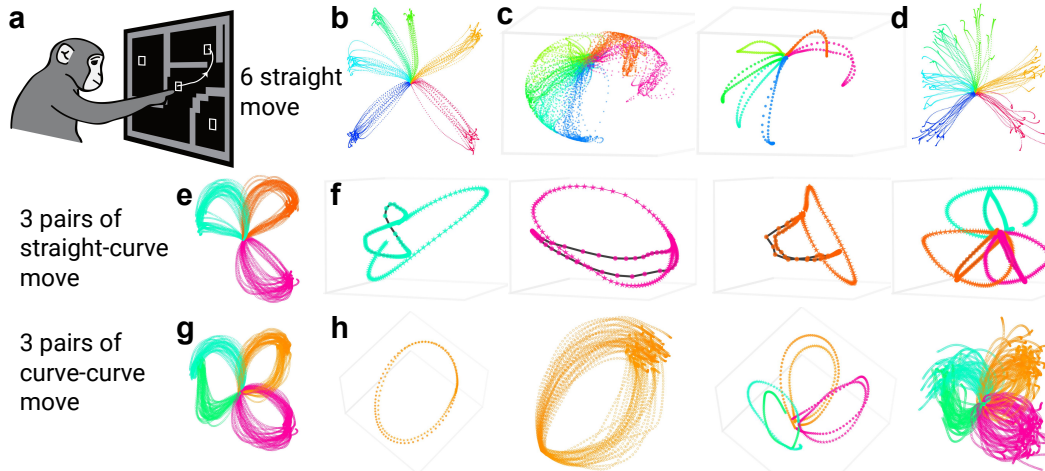

Figure 8: Distinct latent dynamics align with different movements in M1. **a** A monkey performs a curved reaching task through a virtual maze while neural activity from M1 is recorded simultaneously. The task includes multiple reaching directions, and the monkey makes curved movements when encountering a barrier on the trajectory (reproduced from [22]). **b** Hand positions for six target directions without barriers (i.e., straight movements). **c** Latent dynamics from a single trial (left) and averaged across trials (right). **d** Explained variance of hand movements (velocity: 0.79, position: 0.92). Hand directions (represented by colors) are assigned manually. **e** Hand movements for three target directions, both without barriers (straight movements) and with barriers (curved movements). **f** Latent dynamics of curved (stars) and straight (dots with black line) hand movements shown separately and combined across directions. **g** Hand movements for three target directions, all with barriers (curved movements). **h** Latent dynamics trained on one pair (1st) or three pairs (3rd) of curved movements, with decoder-explained hand movements (2nd and 4th). Fig. 17 provides a comparison of results with CEBRA.

dynamics under various conditions. For BMI applications, we demonstrate that NER, combined with a simple linear decoder, can predict hand movements across years, brain areas, and hemispheres. This capability enables training on latent dynamics within and between subjects, allowing for movement prediction in different subjects. The linear decoder's lack of hyperparameters is also an advantage.

The application of NER is not limited to hand movements using neurophysiological recordings. This includes latent dynamics in the hippocampus representing the body position of running rats and latent dynamics in the visual cortex representing embedded video features [27]. Similarly, recording modalities are not limited to single-neuron electrophysiology; other methods, such as calcium imaging, local-field potentials, and EEG, can also be used.

In our final experiments with curved movements, we manually selected three pairs of reaching tasks that involved curved trajectories. Both NER and CEBRA failed when all 108 movement conditions were trained simultaneously. Furthermore, beyond the straight and curved movements examined here in macaque monkeys, more complex movements, such as handwriting [29] and speech [29, 20], exist primarily in humans. These movements are also continuous but may require more than a 3D latent space for effective representation, and ranking the distances of complex movements adds further challenges. Nevertheless, uncovering the latent dynamics underlying these complex movements, which has yet to be achieved[7], could greatly advance BMI applications [3, 6].

## Acknowledgments and Disclosure of Funding

We greatly appreciate the Miller and Shenoy labs for publicly releasing their experimental data on macaque monkeys. We also thank the Mathis lab for the CEBRA code, which served as the basis for our NER. This work was supported by National Institute of Deafness and Other Communications Disorders grant DC003180 (X.W.). No conflicts of interest are declared by the authors.

# References

[1] Mehdi Azabou, Mohammad Gheshlaghi Azar, Ran Liu, Chi-Heng Lin, Erik C Johnson, Kiran Bhaskaran-Nair, Max Dabagia, Bernardo Avila-Pires, Lindsey Kitchell, Keith B Hengen, et al. Mine your own view: Self-supervised learning through across-sample prediction. *arXiv preprint arXiv:2102.10106*, 2021.

[2] Matteo Carandini and David J Heeger. Normalization as a canonical neural computation. *Nature reviews neuroscience*, 13(1):51–62, 2012.

[3] Nicholas S Card, Maitreyee Wairagkar, Carrina Iacobacci, Xianda Hou, Tyler Singer-Clark, Francis R Willett, Erin M Kunz, Chaofei Fan, Maryam Vahdati Nia, Darrel R Deo, et al. An accurate and rapidly calibrating speech neuroprosthesis. *New England Journal of Medicine*, 391(7):609–618, 2024.

[4] Le Chang and Doris Y Tsao. The code for facial identity in the primate brain. *Cell*, 169(6):1013–1028, 2017.

[5] Mark M Churchland, John P Cunningham, Matthew T Kaufman, Justin D Foster, Paul Nuyujukian, Stephen I Ryu, and Krishna V Shenoy. Neural population dynamics during reaching. *Nature*, 487(7405):51–56, 2012.

[6] Chaofei Fan, Nick Hahn, Foram Kamdar, Donald Avansino, Guy Wilson, Leigh Hochberg, Krishna V Shenoy, Jaimie Henderson, and Francis Willett. Plug-and-play stability for intracortical brain-computer interfaces: a one-year demonstration of seamless brain-to-text communication. *Advances in neural information processing systems*, 36, 2024.

[7] Cátia Fortunato, Jorge Bennasar-Vázquez, Junchol Park, Joanna C Chang, Lee E Miller, Joshua T Dudman, Matthew G Perich, and Juan A Gallego. Nonlinear manifolds underlie neural population activity during behaviour. *bioRxiv*, 2023.

[8] Winrich A Freiwald, Doris Y Tsao, and Margaret S Livingstone. A face feature space in the macaque temporal lobe. *Nature neuroscience*, 12(9):1187, 2009.

[9] Juan A Gallego, Matthew G Perich, Raeed H Chowdhury, Sara A Solla, and Lee E Miller. Long-term stability of cortical population dynamics underlying consistent behavior. *Nature neuroscience*, 23(2):260–270, 2020.

[10] Juan A Gallego, Matthew G Perich, Lee E Miller, and Sara A Solla. Neural manifolds for the control of movement. *Neuron*, 94(5):978–984, 2017.

[11] Juan A Gallego, Matthew G Perich, Stephanie N Naufel, Christian Ethier, Sara A Solla, and Lee E Miller. Cortical population activity within a preserved neural manifold underlies multiple motor behaviors. *Nature communications*, 9(1):4233, 2018.

[12] Cecilia Gallego-Carracedo, Matthew G Perich, Raeed H Chowdhury, Lee E Miller, and Juan Álvaro Gallego. Local field potentials reflect cortical population dynamics in a region-specific and frequency-dependent manner. *Elife*, 11:e73155, 2022.

[13] Joshua I Glaser, Ari S Benjamin, Raeed H Chowdhury, Matthew G Perich, Lee E Miller, and Konrad P Kording. Machine learning for neural decoding. *Eneuro*, 7(4), 2020.

[14] Cole Hurwitz, Akash Srivastava, Kai Xu, Justin Jude, Matthew Perich, Lee Miller, and Matthias Hennig. Targeted neural dynamical modeling. *Advances in Neural Information Processing Systems*, 34:29379–29392, 2021.

[15] Mohammad Reza Keshtkaran, Andrew R Sedler, Raeed H Chowdhury, Raghav Tandon, Diya Basrai, Sarah L Nguyen, Hansem Sohn, Mehrdad Jazayeri, Lee E Miller, and Chethan Pandarinath. A large-scale neural network training framework for generalized estimation of single-trial population dynamics. *Nature Methods*, 19(12):1572–1577, 2022.

[16] Dmitry Kobak, Wieland Brendel, Christos Constantinidis, Claudia E Feierstein, Adam Kepecs, Zachary F Mainen, Xue-Lian Qi, Ranulfo Romo, Naoshige Uchida, and Christian K Machens. Demixed principal component analysis of neural population data. *elife*, 5:e10989, 2016.

[17] Eric Kenji Lee, Hymavathy Balasubramanian, Alexandra Tsolias, Stephanie Udochukwu Anakwe, Maria Medalla, Krishna V Shenoy, and Chandramouli Chandrasekaran. Non-linear dimensionality reduction on extracellular waveforms reveals cell type diversity in premotor cortex. *Elife*, 10:e67490, 2021.

[18] Ran Liu, Mehdi Azabou, Max Dabagia, Chi-Heng Lin, Mohammad Gheshlaghi Azar, Keith Hengen, Michal Valko, and Eva Dyer. Drop, swap, and generate: A self-supervised approach for generating neural activity. *Advances in neural information processing systems*, 34:10587–10599, 2021.

[19] Leland McInnes, John Healy, and James Melville. Umap: Uniform manifold approximation and projection for dimension reduction. *arXiv preprint arXiv:1802.03426*, 2018.

[20] Sean L Metzger, Kaylo T Littlejohn, Alexander B Silva, David A Moses, Margaret P Seaton, Ran Wang, Maximilian E Dougherty, Jessie R Liu, Peter Wu, Michael A Berger, et al. A high-performance neuroprosthesis for speech decoding and avatar control. *Nature*, 620(7976):1037–1046, 2023.

[21] Chethan Pandarinath, Daniel J O'Shea, Jasmine Collins, Rafal Jozefowicz, Sergey D Stavisky, Jonathan C Kao, Eric M Trautmann, Matthew T Kaufman, Stephen I Ryu, Leigh R Hochberg, et al. Inferring single-trial neural population dynamics using sequential auto-encoders. *Nature methods*, 15(10):805–815, 2018.

[22] Felix Pei, Joel Ye, David Zoltowski, Anqi Wu, Raeed H Chowdhury, Hansem Sohn, Joseph E O'Doherty, Krishna V Shenoy, Matthew T Kaufman, Mark Churchland, et al. Neural latents benchmark'21: evaluating latent variable models of neural population activity. *arXiv preprint arXiv:2109.04463*, 2021.

[23] Matthew G Perich, Juan A Gallego, and Lee E Miller. A neural population mechanism for rapid learning. *Neuron*, 100(4):964–976, 2018.

[24] Neil C Rabinowitz, Ben DB Willmore, Jan WH Schnupp, and Andrew J King. Contrast gain control in auditory cortex. *Neuron*, 70(6):1178–1191, 2011.

[25] Mostafa Safaie, Joanna C Chang, Junchol Park, Lee E Miller, Joshua T Dudman, Matthew G Perich, and Juan A Gallego. Preserved neural dynamics across animals performing similar behaviour. *Nature*, 623(7988):765–771, 2023.

[26] Omid G Sani, Hamidreza Abbaspourazad, Yan T Wong, Bijan Pesaran, and Maryam M Shanechi. Modeling behaviorally relevant neural dynamics enabled by preferential subspace identification. *Nature Neuroscience*, 24(1):140–149, 2021.

[27] Steffen Schneider, Jin Hwa Lee, and Mackenzie Weygandt Mathis. Learnable latent embeddings for joint behavioural and neural analysis. *Nature*, 617(7960):360–368, 2023.

[28] Laurens Van der Maaten and Geoffrey Hinton. Visualizing data using t-sne. *Journal of machine learning research*, 9(11), 2008.

[29] Francis R Willett, Donald T Avansino, Leigh R Hochberg, Jaimie M Henderson, and Krishna V Shenoy. High-performance brain-to-text communication via handwriting. *Nature*, 593(7858):249–254, 2021.

[30] Joel Ye and Chethan Pandarinath. Representation learning for neural population activity with neural data transformers. *arXiv preprint arXiv:2108.01210*, 2021.

[31] Kaiwen Zha, Peng Cao, Jeany Son, Yuzhe Yang, and Dina Katabi. Rank-n-contrast: Learning continuous representations for regression. *Advances in Neural Information Processing Systems*, 36, 2024.

[32] Ding Zhou and Xue-Xin Wei. Learning identifiable and interpretable latent models of high-dimensional neural activity using pi-vae. *Advances in Neural Information Processing Systems*, 33:7234–7247, 2020.

[33] Feng Zhu, Harrison A Grier, Raghav Tandon, Changjia Cai, Anjali Agarwal, Andrea Giovannucci, Matthew T Kaufman, and Chethan Pandarinath. A deep learning framework for inference of single-trial neural population dynamics from calcium imaging with subframe temporal resolution. *Nature neuroscience*, 25(12):1724–1734, 2022.

# A  Appendix / supplemental material

## A.1  Computer

Operating System is Ubuntu 22.04.3 LTS, computer memory is 42 GB, CPU is Intel Xeon W-2225, and GPU is NVIDIA RTX A5000.

## A.2  Center-out reaching experiments in M1, PMd, and S1

### A.2.1  Behavior

The dataset comprises behavioral task data from two male Macaca mulatta monkeys (Monkeys H and C). These monkeys were trained to sit in a primate chair and perform a center-out reaching task using a planar manipulandum with the hand contralateral to the implanted hemisphere. During each trial, the monkey started by moving a cursor to a central target. After a variable waiting period, one of eight outer targets (equally spaced along a circle of 6–8 cm radius) was presented. Monkeys C and H differed in the task protocols:

**Monkey C**: Trained to wait for an auditory go cue during a delay period of 0.5–1.5 seconds while the target remained visible. Upon receiving the cue, the monkey had to move the cursor to the outer target within 1 second and hold it there for 0.5 seconds to receive a liquid reward.

**Monkey H**: No delay period; the monkey had to move the cursor to the outer target within 1 second and hold it there for 0.1 seconds.

For both monkeys, the trial restarted by returning the cursor to the central target. Endpoint positions of the manipulandum were recorded at 1 kHz, and task event timings were digitally logged. Hand velocity was computed as the derivative of hand position. The dataset includes 6 sessions for Monkey C and 5 sessions for Monkey H, considering only successful trials (an average of 307±221 trials per session, mean ± s.d.)

**Neural recordings**

The dataset consists of neural recordings from two male Macaque monkeys. These recordings were obtained using 96-channel Utah electrode arrays implanted in specific cortical regions.

**Monkey C**: Initially implanted in the right primary motor cortex (M1) and later received implants in the left M1 and dorsal premotor cortex (PMd) (denoted as CR and CL, respectively).

**Monkey H**: Implanted in area 2 of the primary somatosensory cortex of the left hemisphere.

Neural activity was recorded using a Cerebus system (Blackrock Microsystems, Salt Lake City, UT) at a sampling frequency of 30 kHz. The recorded signals underwent band-pass filtering (250–5000 Hz) and were converted to spike times based on threshold crossings. Spike sorting was performed using specialized software (Offline Sorter v3, Plexon, Inc, Dallas, TX) to identify putative neurons.

| Date | Monkey | Hemisphere | Trial | M1 | PMd | S1 |
|---|---|---|---|---|---|---|
| 150313 | Chewie | Right | 1038 | 86 | n/a | n/a |
| 150309 | Chewie | Right | 1026 | 72 | n/a | n/a |
| 150629 | Chewie | Right | 179 | 49 | n/a | n/a |
| 150630 | Chewie | Right | 178 | 44 | n/a | n/a |
| 160929 | Chewie | Left | 208 | 74 | 114 | n/a |
| 161005 | Chewie | Left | 202 | 82 | 167 | n/a |
| 161006 | Chewie | Left | 209 | 63 | 192 | n/a |
| 161007 | Chewie | Left | 168 | 70 | 137 | n/a |
| 161014 | Chewie | Left | 740 | 88 | 190 | n/a |
| 161021 | Chewie | Left | 286 | 84 | 211 | n/a |
| 171201 | Han | Left | 292 | n/a | n/a | 70 |
| 171204 | Han | Left | 255 | n/a | n/a | 83 |
| 171207 | Han | Left | 245 | n/a | n/a | 72 |

Table 1: Datasets for the center-out reaching experiments.

**Datasets**

All the center-out reaching experiments using the open source data from: `https://datadryad.org/stash/dataset/doi:10.5061/dryad.xd2547dkt` This data is released accompanying this paper:`https://elifesciences.org/articles/73155#data` We used all sessions from Monkey Chewie and Monkey Han. We chose these Monkeys because one session in Chewie is used by piVAE paper[32] and one session in Monkey Han is used by CEBRA paper[27]. Although the datasets come from same session in same Monkey, the temporal resolution is much higher for the datasets used by piVAE and CEBRA papers. The data is Matlab format and we extract following information: tgtDir (Target direction, radians for Monkey Chewie and degrees for Monkey Han), idx-goCueTime (The time go Cue is one), vel(XY velocities), M1-spikes for both Chewie 2015 and Chewie 2016, and PMd-spikes only for Chewie 2016. The time bin is 30ms and we extract all the spikes after each go Cue. We extracted 40 bins for Monkey Chewie and 35 bins for Monkey Han, because most trial in Monkey Han has short acquisition window than 40 bins (afte go Cue). We smoothed the discrete spike count in the Matlab using a Gaussian kernel. The standard deviation is 1.5 and kernel size is six standard deviations. We keep all the trials and neurons. The number of trials and neurons are shown in Table 1. Our NER is just a modification of the loss function used by CEBRA: `https://github.com/AdaptiveMotorControlLab/CEBRA` The RNC loss could be downloaded from: `https://github.com/kaiwenzha/Rank-N-Contrast` The iterations is 20000, learning rate is 1e-4, and batch size 512. The temperature is fixed to 1 for both NER and CEBRA. The output dimension of NER is fixed to 3. For CEBRA, we used output dimension of 3 for visualizing the latent dynamics and linear models decoding. We only used 16 dimensional embeddings for k-NN decoders. For the piVAE, we did not use the original version which is based on older version of Tensorflow. `https://github.com/zhd96/pi-vae` Instead, we used the modified conv-pi-VAE that is already included into the CEBRA package. We fixed the random seed to 42, and using batch size of 200 and iterations of 300.

### A.2.2  Curved hand movements experiments in M1 Fig 8 17

The MC_Maze dataset includes recordings from the primary motor and dorsal premotor cortices of a monkey performing reaches to visual targets in a virtual maze with an instructed delay. This dataset comprises 108 different task configurations, each varying in target positions, barrier numbers, and barrier positions. The monkey repeated each task configuration multiple times in random order, resulting in 2,869 trials recorded in a single session with 182 neurons and simultaneous hand kinematics monitoring. This datasets could be downloaded from: `https://dandiarchive.org/dandiset/000128`, `https://github.com/dandisets/000128`. In our works, we used the NLB21 package[22] to download the data from DANDI: `https://github.com/neurallatents` Our code (Jupyter Notebook) is modified from: `https://github.com/neurallatents/neurallatents.github.io/blob/master/notebooks/mc_maze.ipynb` We used a time bin of 5ms (raw resolution is 1ms) and Gaussian window of 50ms. For six straight movements, we only use version 0 and trial type of 13, 29, 17, 38, 6, 18. For the straight-curved and curved-curved movements, we keep all three versions of task (one straight and two curved). We removed the target angle during training. We used trial type of 13, 38, 18 for straight-curved and 37,1, 31,38, 34,18 for curved-curved movements. The iterations is 5000, learning rate is 1e-4, and batch size 512.

### A.3  Decoders

We rotate the latent dynamics with reference to same target before decoding. We used orthogonal Procrustes from scipy for this purpose: `https://docs.scipy.org/doc/scipy/reference/generated/scipy.linalg.orthogonal_procrustes.html`. We picked on target angle and rotate the whole 3D latent dynamics using the computed orthogonal matrix. This rotation will not modify local detail or the relative positions of each reaching direction. For the linear decoders using linear and logistic regression, both are imported from "linear_model" of sklearn. There is no parameter or hyperparameter for the linear regression model. For the logistic regression model, we used the following parameters: max_iter=500, multi_class='multinomial', solver='lbfgs' For the k-nearest neighbors Regressor and Classifier, they are both imported from "neighbors" in sklearn. We used "GridSearchCV" in sklearn to searach the best "n_neighbors" range from 2 to 1024.

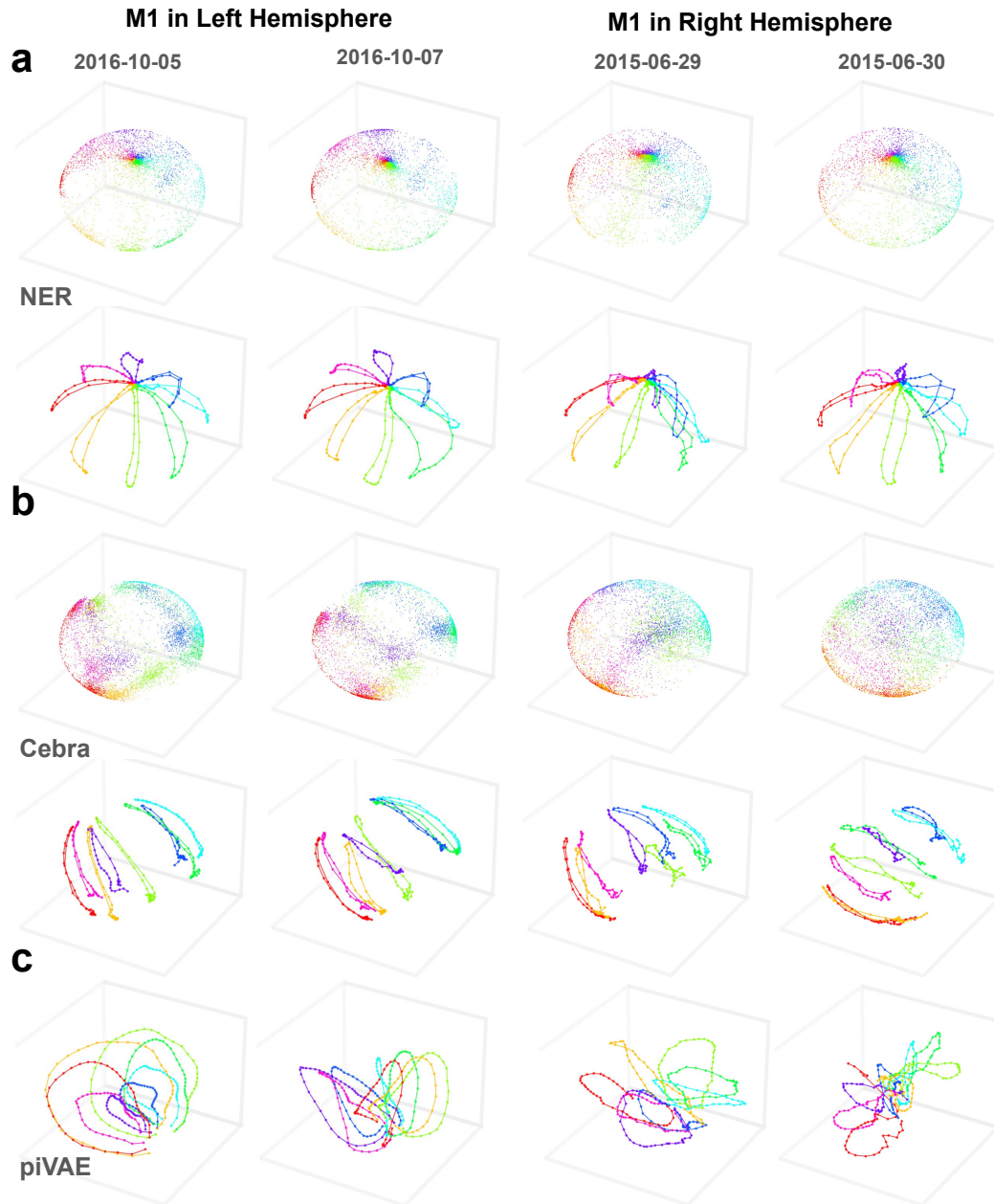

Figure 9: NER reveals consistent and movements aligned latent dynamics in M1 for the remaining four sessions. Extra four sessions' latent dynamics at left and right hemisphere of Monkey C after rotating relative to target session in Fig 3 (2016-10-14).

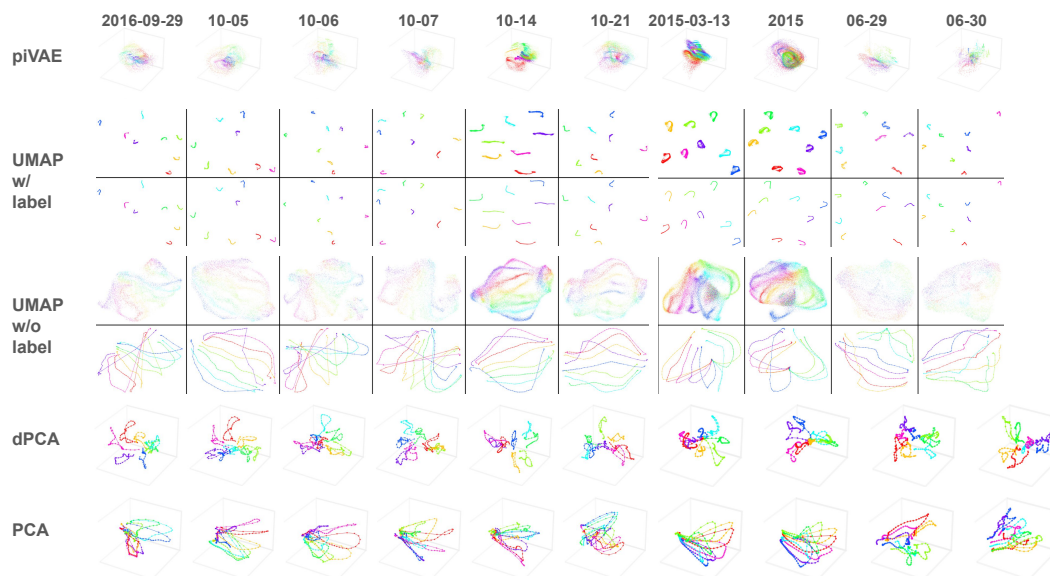

Figure 10: Neural embeddings revealed by five other dimensionality reduction methods. Single trial (top) and trial averaged (bottom) latent dynamics revealed by five other dimensionality reduction methods.

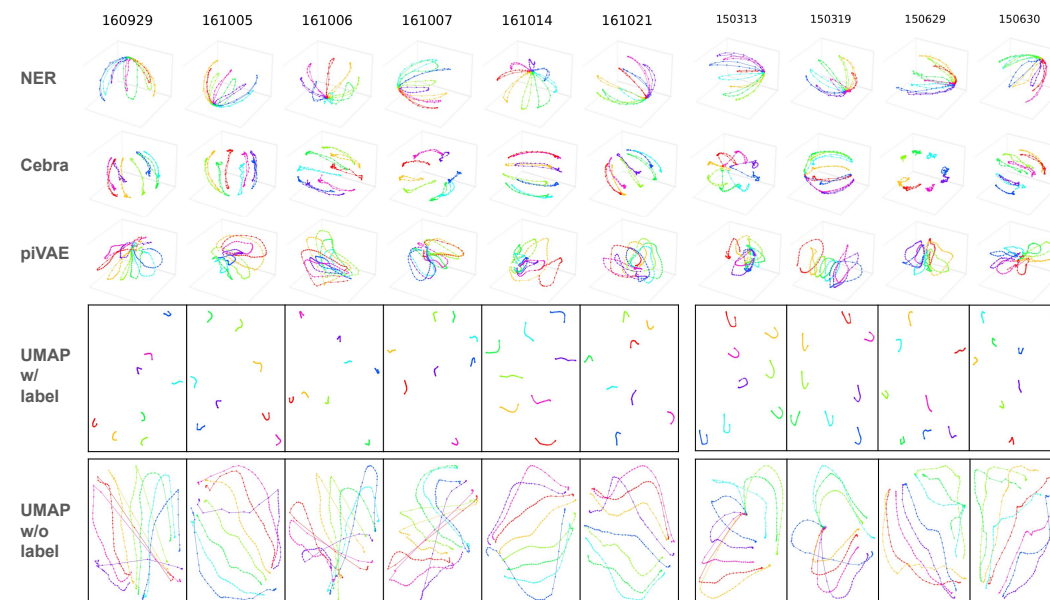

Figure 11: Latent dynamics without rotation. Trial-averaged latent dynamics revealed by five dimensionality reduction methods without rotation.

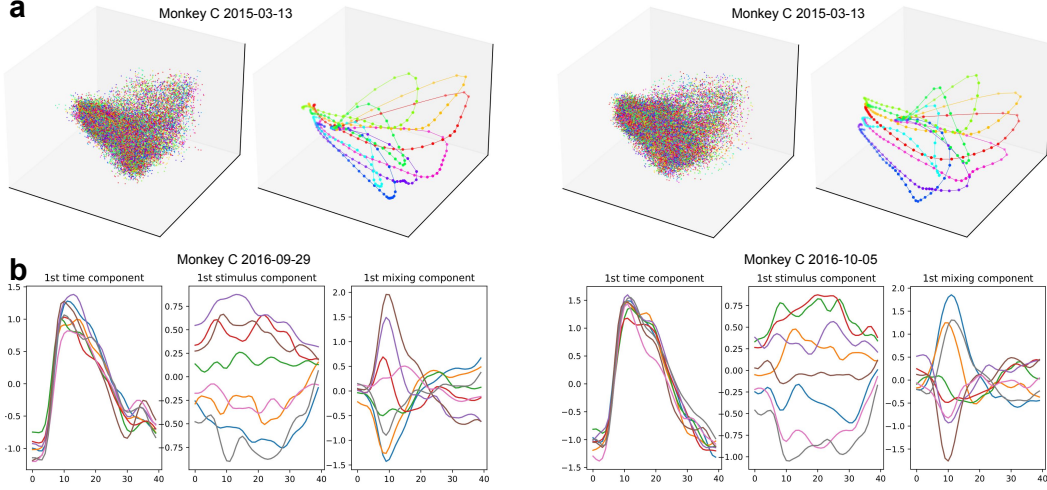

Figure 12: Mixed single trial latent dynamics in PCA and time-stimulus components revealed by dPCA. **a** Unlike other five dimensionality reduction methods, single trial latent dynamics revealed by principal component analysis (PCA) and demixed PCA (dPCA) is mixed and latency dynamics are only identifiable after averaging. **b** In the trial-averaged latent dynamics, dPCA reveals three components (left, middle, right) at eight directions (different colors): time component aligned with go cue regardless of directions, separated stimulus component varied across time, and mixed component aligned with go cue and different for each direction.

Table 2: Statistical analysis and quantitative comparisons of the presented results. Figs. 3 and 6a show the correlation coefficients between each pair of averaged latent dynamics. Here, "d." refers to values on the diagonal, "o." refers to values off the diagonal, "L" denotes the left panel, and "R" denotes the right panel. The 2nd to 4th columns display the mean and standard deviation, while the 5th to 7th columns provide the t-statistics and p-values.

| Fig. | NER | CEBRA | piVAE | NER vs CE | NER vs pi | CE vs pi |
|---|---|---|---|---|---|---|
| 3 | 0.95, 0.03 | 0.92, 0.05 | 0.43, 0.25 | 7.9, 5.5e-10 | 15.3, 2.9e-19 | 14.8, 1.0e-18 |
| 5a-d | 0.71, 0.1 | -3.08, 0.6 | -1.21, 1.4 | 16.8, 4.3e-08 | 4.1, 2.5e-03 | -5.8, 2.8e-04 |
| 5a-o | 0.64, 0.1 | -3.22, 1.0 | -2.05, 1.6 | 40.9, 1.7e-59 | 16.3, 1.7e-28 | -9.5, 3.9e-15 |
| 5b-d | 0.76, 0.1 | 0.75, 0.1 | 0.87, 0.1 | 0.4, 7.0e-01 | -3.5, 7.2e-03 | -3.2, 1.1e-02 |
| 5b-o | 0.70, 0.1 | -1.07, 1.5 | -1.05, 1.1 | 11.5, 3.3e-19 | 15.4, 6.4e-27 | -0.1, 8.9e-01 |
| 5c-d | 0.94, 0.0 | 0.74, 0.0 | 0.67, 0.2 | 11.6, 9.9e-07 | 4.4, 1.8e-03 | 1.4, 1.9e-01 |
| 5c-o | 0.88, 0.1 | 0.71, 0.1 | 0.23, 0.4 | 19.2, 2.5e-33 | 17.9, 3.0e-31 | 12.6, 1.4e-21 |
| 5d-d | 0.93, 0.0 | 0.93, 0.0 | 0.96, 0.0 | 0.5, 6.6e-01 | -3.7, 5.0e-03 | -3.1, 1.2e-02 |
| 5d-o | 0.89, 0.0 | 0.18, 0.5 | 0.13, 0.4 | 13.1, 2.0e-22 | 16.5, 8.0e-29 | 0.7, 4.8e-01 |
| 5e-d | 93, 4.4 | 90, 4.6 | 81, 11.9 | 2.5, 3.6e-02 | 2.4, 4.0e-02 | 1.9, 9.5e-02 |
| 5e-o | 90, 5.9 | 79, 9.0 | 43, 15.2 | 13.6, 1.7e-23 | 26.0, 2.6e-43 | 17.5, 1.2e-30 |
| 5f-d | 92, 4.1 | 93, 4.2 | 98, 1.5 | -0.3, 7.5e-01 | -4.9, 8.3e-04 | -3.4, 7.5e-03 |
| 5f-o | 89, 5.8 | 57, 16.3 | 44, 15.9 | 19.1, 3.2e-33 | 25.7, 5.4e-43 | 5.6, 2.5e-07 |
| 6a | 0.87, 0.08 | 0.81, 0.10 | 0.56, 0.09 | 14.8, 1.1e-18 | 29.3, 1.6e-30 | 20.7, 2.8e-24 |
| 6bL-d | 0.75, 0.0 | -3.06, 0.4 | -1.56, 1.1 | 35.2, 1.2e-12 | 7.2, 1.8e-05 | -4.8, 5.8e-04 |
| 6bL-o | 0.74, 0.0 | -3.10, 0.3 | -2.78, 2.2 | 124.6, 5.7e-138 | 18.2, 9.7e-38 | -1.6, 1.1e-01 |
| 6cL-d | 0.78, 0.1 | 0.78, 0.1 | 0.84, 0.1 | -0.0, 9.8e-01 | -2.4, 3.3e-02 | -1.9, 7.7e-02 |
| 6cL-o | 0.76, 0.1 | -0.72, 1.0 | -0.62, 0.8 | 17.7, 1.8e-36 | 19.8, 3.5e-41 | -0.9, 3.7e-01 |
| 6bR-d | 0.94, 0.0 | 0.75, 0.0 | 0.58, 0.1 | 36.3, 8.4e-13 | 9.5, 1.3e-06 | 4.5, 9.7e-04 |
| 6bR-o | 0.93, 0.0 | 0.74, 0.0 | 0.21, 0.3 | 77.2, 4.3e-111 | 24.3, 2.6e-50 | 18.2, 1.1e-37 |
| 6cR-d | 0.93, 0.0 | 0.94, 0.0 | 0.95, 0.0 | -0.1, 9.3e-01 | -1.2, 2.4e-01 | -1.0, 3.4e-01 |
| 6cR-o | 0.92, 0.0 | 0.25, 0.5 | 0.18, 0.5 | 15.3, 5.6e-31 | 17.2, 2.6e-35 | 1.1, 2.8e-01 |
| 7d-d | 0.67, 0.0 | -4.59, 1.3 | -0.38, 0.3 | 5.6, 3.0e-02 | 5.9, 2.8e-02 | -5.5, 3.2e-02 |
| 7d-o | 0.56, 0.1 | -4.79, 1.1 | -0.75, 0.1 | 11.4, 9.0e-05 | 19.4, 6.7e-06 | -8.3, 4.1e-04 |
| 7e-d | 0.90, 0.0 | 0.60, 0.0 | 0.73, 0.1 | 18.5, 2.9e-03 | 3.1, 8.9e-02 | -1.8, 2.2e-01 |
| 7e-o | 0.77, 0.1 | 0.45, 0.1 | 0.51, 0.1 | 5.9, 1.9e-03 | 3.3, 2.1e-02 | -0.9, 4.1e-01 |

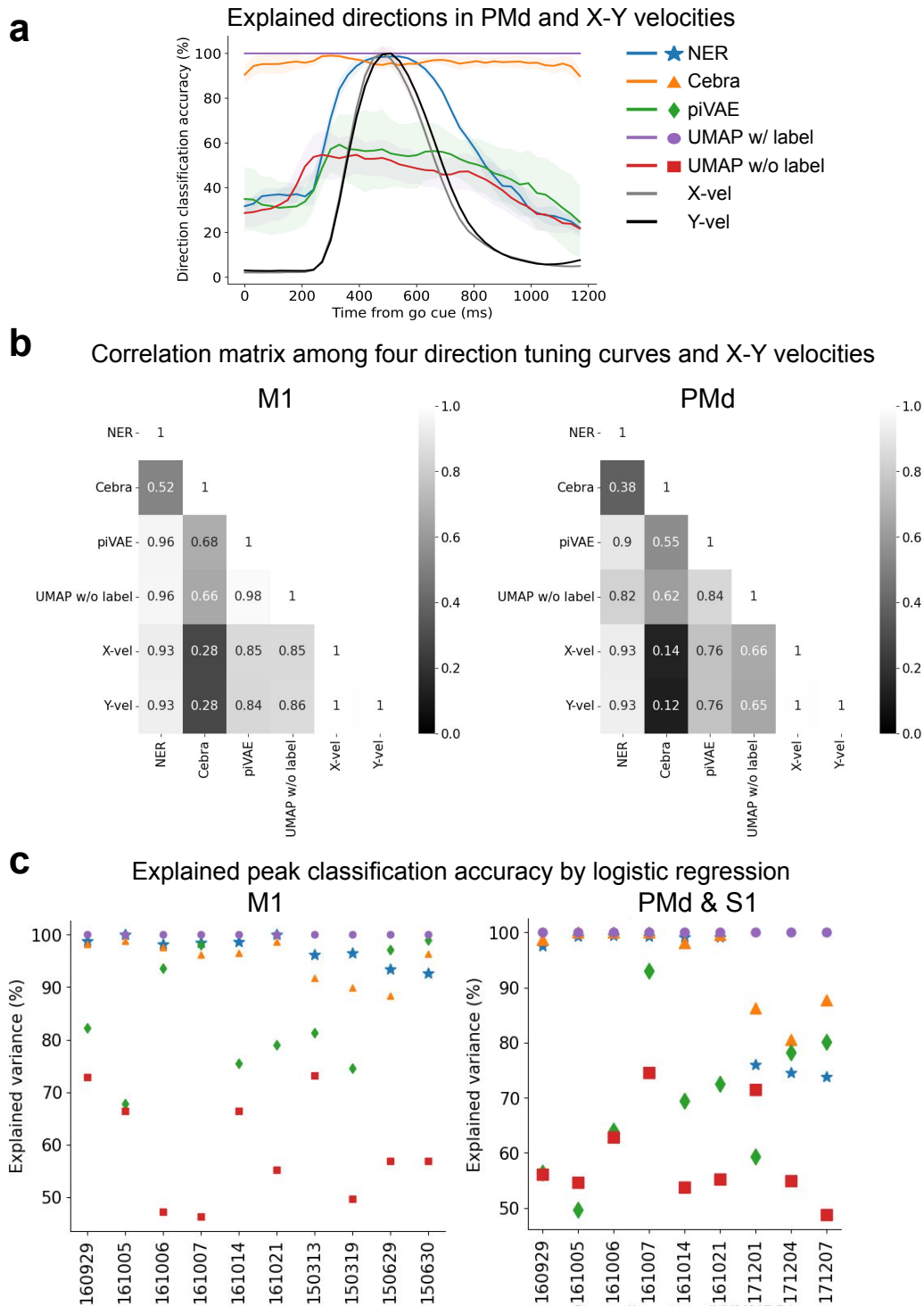

Figure 13: Hand directions tunings and explained peak classification accuracy of hand directions in M1, PMd, and S1. **a** Hand directions explained accuracy using a logistic regression models trained on the latent dynamics revealed by five dimensionality reduction methods. Shaded areas are standard deviation over six sessions from PMd in the Monkey C. Notice the only NER reveal hand velocities dependent direction tuning curves that peak around 500 ms. **b** Correlation coefficients matrix between direction tuning curves and velocities in M1 (left) and PMd (right). **c** Explained variance on hand directions using logistic regression decoder.

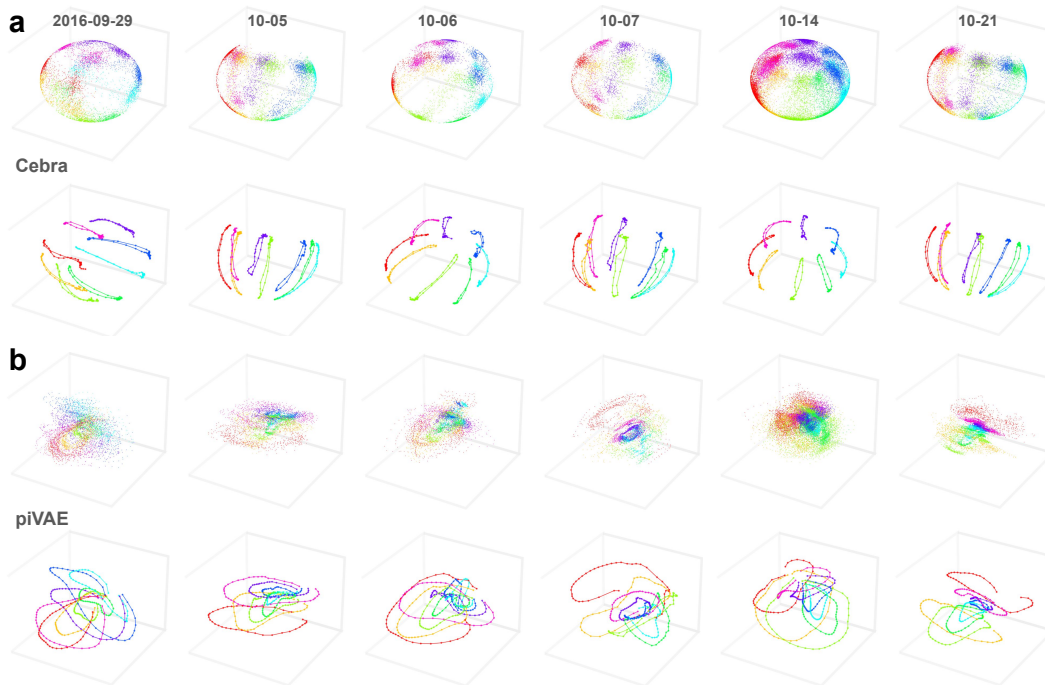

Figure 14: Latent dynamics in PMd revealed by CEBRA and piVAE. Single trial and trial averaged latent dynamics revealed by CEBRA and piVAE. All the figures are rotated with reference with the one session shown in Fig 3 (2016-10-14).

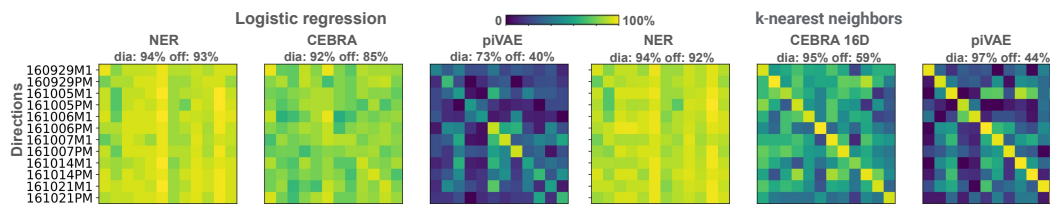

Figure 15: Direction decoding accuracy. Same date, cross date, and cross brain areas decoding of hand directions using linear regression (left) and k-nearest neighbors decoder trained on the latent dynamics revealed by three methods. Notice the range of color bars is 0-100 for all six figures.

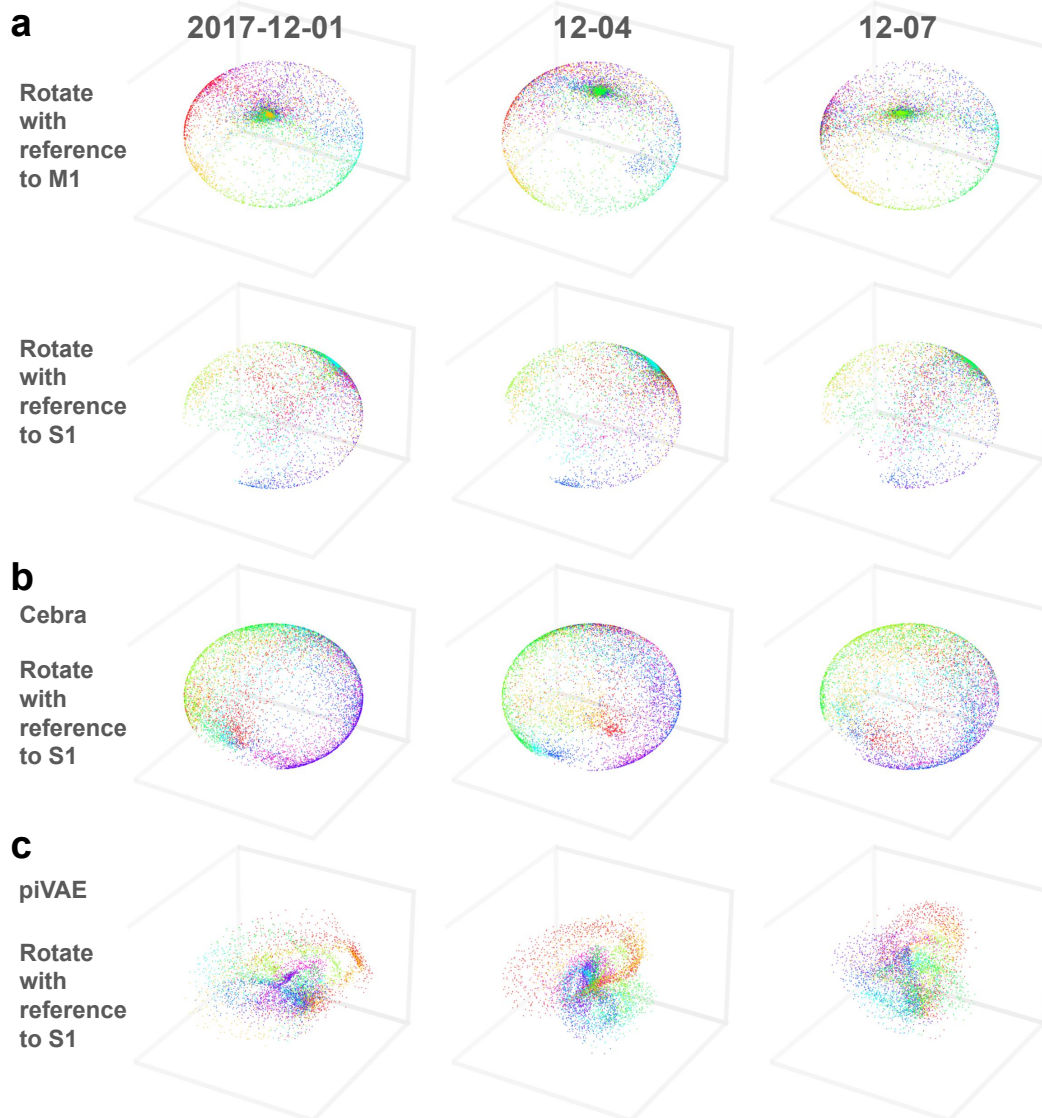

Figure 16: Latent dynamics in S1 revealed by CEBRA and piVAE. **a** Top, single trial latent dynamics revealed by CEBRA are rotated with reference to the one session in M1. Bottom, latent dynamis are rotated with reference to the first session in S1. **b** Similar to the bottom figure of a but using CEBRA dimensionality reduction method. **c** Similar to c but using piVAE dimensionality reduction method.

3 pairs of straight-curve move

3 pairs of curve-curve move

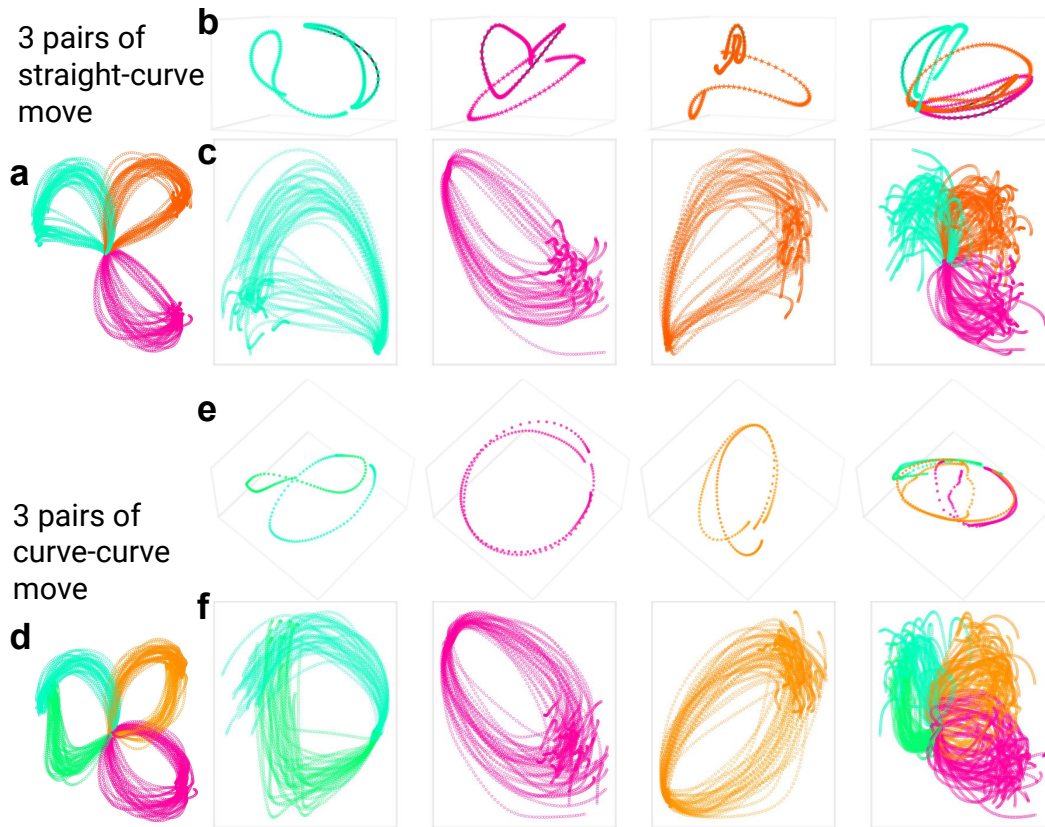

Figure 17: Distinct latent dynamics revealed by CEBRA. Similar to Fig 7 but using the latent dynamics revealed by CEBRA. **a** Ground truth motion trajectories at three directions for straight-curve movements. **b** Latency dynamics trained on three directions separately and combined. Notice the latent dynamics of straight hand movements (dots and black line) are either mixed (1st) with curved movements or squeezed (3rd). Latent dynamics trained by three directions combined are overlapped. **c** Linear models predicted hand trajectories. The directions are manually assigned. The explained variance for hand velocities (directions) are 87% (94%), 83% (93%), 82% (94%), and 66% (86%). Notice all the values are lower than NER which are 92% (95%), 89% (96%), 90% (96%), and 81% (91%), especially for the combined directions. **d** Ground truth motion trajectories at four directions for paired curve-curve movements. **e** Latency dynamics trained on three directions separately and combined. Notice latent dynamics trained by three directions combined are overlapped. **f** Linear models predicted hand trajectories. The directions are manually assigned. The explained variance for hand velocities (directions) are 85% (93%), 81% (92%), 85% (94%), and 61% (80%). Notice all the values are lower than NER which are 87% (95%), 89% (95%), 91% (96%), and 81% (91%), especially for the combined directions.

